# Spiral Waves in Integrate–and–Fire Neural Networks

**John G. Milton**
Department of Neurology
The University of Chicago
Chicago, IL 60637

**Po Hsiang Chu**
Department of Computer Science
DePaul University
Chicago, IL 60614

**Jack D. Cowan**
Department of Mathematics
The University of Chicago
Chicago, IL 60637

## Abstract

The formation of propagating spiral waves is studied in a randomly connected neural network composed of integrate–and–fire neurons with recovery period and excitatory connections using computer simulations. Network activity is initiated by periodic stimulation at a single point. The results suggest that spiral waves can arise in such a network via a sub–critical Hopf bifurcation.

## 1   Introduction

In neural networks activity propagates through populations, or layers, of neurons. This propagation can be monitored as an evolution of spatial patterns of activity. Thirty years ago, computer simulations on the spread of activity through 2–D randomly connected networks demonstrated that a variety of complex spatio–temporal patterns can be generated including target waves and spirals (Beurle, 1956, 1962; Farley and Clark, 1961; Farley, 1965). The networks studied by these investigators correspond to inhomogeneous excitable media in which the probability of interneuronal connectivity decreases exponentially with distance. Although travelling spiral waves can readily be formed in excitable media by the introduction of non–uniform

initial conditions (e.g. Winfree, 1987), this approach is not suitable for the study and classification of the dynamics associated with the onset of spiral wave formation. Here we show that spiral waves can "spontaneously" arise from target waves in a neural network in which activity is initiated by periodic stimulation at a single point. In particular, the onset of spiral wave formation appears to occur via a sub–critical Hopf bifurcation.

## 2   Methods

Computer simulations were used to simulate the propagation of activity from a centrally placed source in a neural network containing $100 \times 100$ neurons arranged on a square lattice with excitatory interactions. At $t = 0$ all neurons were at rest except the source. There were free boundary conditions and all simulations were performed on a SUN SPARC 1+ computer.

The network was constructed by assuming that the probability, $\lambda$, of interneuronal connectivity was an exponential decreasing function of distance, i.e.

$$\lambda = \beta \exp(-\alpha |r|)$$

where $\alpha = 0.6, \beta = 1.5$ are constants and $|r|$ is the euclidean interneuronal distance (on average each neuron makes 24 connections and $\sim 1.3$ connections per neuron, i.e. multiple connections occur). Once the connectivity was determined it remained fixed throughout the simulation.

The dynamics of each neuron were represented by an integrate–and–fire model possessing a "leaky" membrane potential and an absolute (1 time step) and relative refractory or recovery period as described previously (Beurle, 1962; Farley, 1965; Farley and Clark, 1961): the membrane and threshold decay constants were, respectively, $k_m = 0.3$ msec$^{-1}$, $k_\theta = 0.03$ msec$^{-1}$. The time step of the network was taken as 1 msec and it was assumed that during this time a neuron transmits excitation to all other neurons connected to it.

## 3   Results

We illustrate the dynamics of a particular network as a function of the magnitude of the excitatory interneuronal excitation, $E$, when all other parameters are fixed. When $E < 0.2$ no activity propagates from the central source. For $0.2 \leq E < 0.58$ target waves regularly emanate from the centrally placed source (Figure 1a). For $E \geq 0.58$ the activity patterns, once established, persisted even when the source was turned off. Complex spiral waves occurred when $0.58 \leq E < 0.63$ (Figures 1b–1d). In these cases spiral meandering, spiral tip break–up and the formation of new spirals (some with multiple arms) occur continuously. Eventually the spirals tend to migrate out of the network. For $E \geq 0.63$ only disorganized spatial patterns occurred without clearly distinguishable wave fronts, except initially (Figures 1e–f).

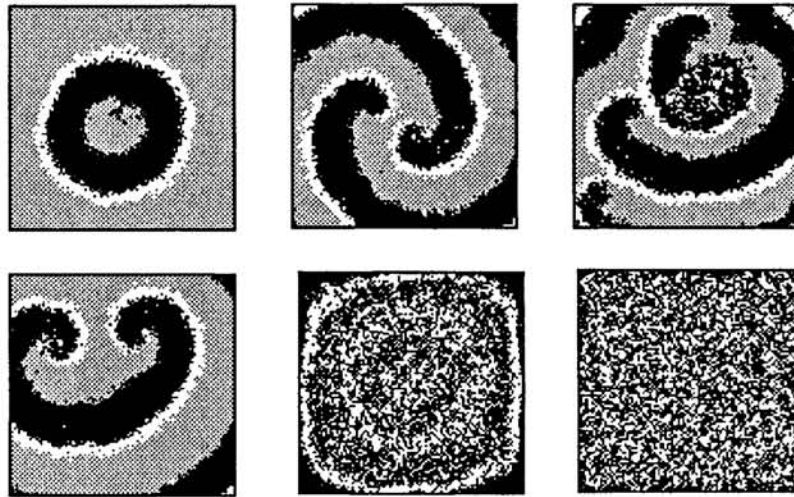

Figure 1: Representative examples of the spatial pattern of neural activity as a function of $E$:(a) $E = 0.45$, (b – e) $E = 0.58$ and (f) $E = 0.72$. Color code: gray = quiescent, white = activated, black = relatively refractory. See text for details.

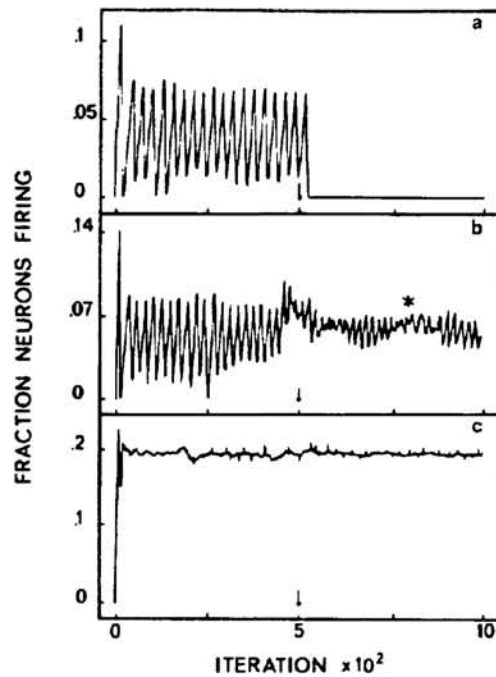

Figure 2: Plot of the fraction of neurons firing per unit time for different values of $E$: (a) 0.45, (b) 0.58, and (c) 0.72. At $t = 0$ all neurons except the central source are quiescent. At $t = 500$ (indicated by ↓) the source is shut off. The region indicated by (∗) corresponds to an epoch in which spiral tip breakup occurs.

The temporal dynamics of the network can be examined by plotting the fraction F of neurons that fire as a function of time. As $E$ is increased through target waves (Figure 2a) to spiral waves (Figure 2b) to disorganized patterns (Figure 2c), the fluctuations in $F$ become less regular, the mean value increases and the amplitude decreases. On closer inspection it can be seen that during spiral wave propagation (Figure 2b) the time series for $F$ undergoes amplitude modulation as reported previously (Farley, 1965). The interval of low amplitude, very irregular fluctuations in $F$ (∗ in Figure 2b) corresponds to a period of spiral tip breakup (Figure 1c).

The appearance of spiral waves is typically preceded by 20–30 target waves. The formation of a spiral wave appears to occur in two steps. First there is an increase in the minimum value of F which begins at $t \sim 420$ and more abruptly occurs at $t \sim 460$ (Figure 2b). The target waves first become asymmetric and then activity propagates from the source region without the more centrally located neurons first entering the quiscent state (Figure 3c). At this time the spatially coherent wave front of the target waves becomes replaced by a disordered noncoherent distribution of active and refractory neurons. Secondly, the dispersed network activity begins to coalesce (Figures 3c and 3d) until at $t \sim 536$ the first identifiable spiral occurs (Figure 3e).

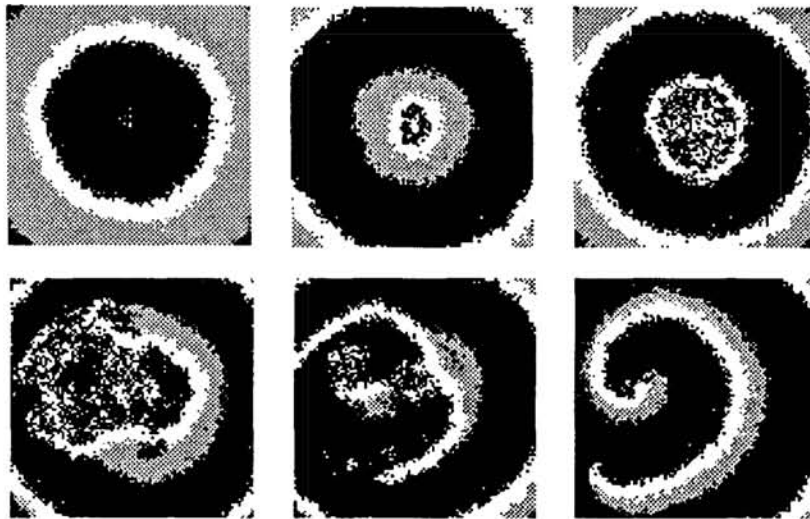

Figure 3: The fraction of neurons firing per unit time, for differing values of generation time t: (a) 175, (b) 345, (c) 465, (d) 503 (e) 536, and (f) 749. At t = 0 all neurons except the central source are quiescent.

It was found that only 4 out of 20 networks constructed with the same $\alpha, \beta$ produced spiral waves for $E = 0.58$ with periodic central point stimulation (simulations, in some cases, ran up to 50,000 generations). However, for all 20 networks, spiral waves could be obtained by the use of non-uniform initial conditions. Moreover, for those networks in which spiral waves occurred, the generation at which they formed differed. These observations emphasize that small fluctuations in the local connectivity of neurons likely play a major role in governing the dynamics of the network.

# 4   Discussion

Self-maintaining spiral waves can arise in an inhomogeneous neural network with uniform initial conditions. Initially well-formed target waves emanate periodically from the centrally placed source. Eventually, provided that E is in a critical range (Figures 1 & 3), the target waves may break up and be replaced by spiral waves. The necessary conditions for spiral wave formation are that: 1) the network be sufficiently tightly connected (Farley, 1965; Farley and Clark, 1961) and 2) the probability of interneuronal connectivity should decrease with distance (unpublished observations). As the network is made more tightly connected the probability that self-maintained activity arises increases provided that E is in the appropriate range (unpublished observations). These criteria are not sufficient to ensure that self-maintained activity, including spiral waves, will form in a given realization of the neural network. It has previously been shown that partially formed spiral–like waves can arise from periodic point stimulation in a model excitable media in which the inhomogeneity arises from a dispersion of refractory times, $k_\theta^{-1}$ (Kaplan, et al, 1988).

Integrate–and–fire neural networks have two stable states: a state in which all neurons are at rest, another associated with spiral waves. Target waves represent a transient response to perturbations away from the stable rest state. Since the neurons have memory (i.e. there is a relative refractory state with $k_\theta \ll k_m$), the mean threshold and membrane potential of the network evolve with time. As a consequence the mean fraction of firing neurons slowly increases (Figure 2b). Our simulations suggest that at some point, provided that the connectivity of the network is suitable, the rest state suddenly becomes unstable and is replaced by a stable spiral wave. This exchange of stability is typical of a sub-critical Hopf bifurcation.

Although complex, but organized, spatio–temporal patterns of spreading activity can readily be generated by a randomly connected neural network, the significance of these phenomena, if any, is not presently clear. On the one hand it is not difficult to imagine that these spatio–temporal dynamics could be related to phenomena ranging from the generation of the EEG, to the spread of epileptic and migraine related activity and the transmission of visual images in the cortex to the formation of patterns and learning by artificial neural networks. On the other hand, the occurence of such phenomena in artificial neural nets could conceivably hinder efficient learning, for example, by slowing convergence. Continued study of the properties of these networks will clearly be necessary before these issues can be resolved.

### Acknowledgements

The authors acknowledge useful discussions with Drs. G. B. Ermentrout, L. Glass and D. Kaplan and financial support from the National Institutes of Health (JM), the Brain Research Foundation (JDC, JM), and the Office of Naval Research (JDC).

### References

R. L. Beurle. (1956) Properties of a mass of cells capable of regenerating pulses. *Phil. Trans. Roy. Soc. Lond.* **240 B**, 55–94.

R. L. Beurle. (1962) Functional organization in random networks. In *Principles of Self-Organization*, H. v. Foerster and G. W. Zopf, eds., pp 291–314. New York, Pergamon Press.

B. G. Farley. (1965) A neuronal network model and the "slow potentials" of electrophysiology. *Comp. in Biomed. Res.* **2**, 265–294.

B. G. Farley & W. A. Clark. (1961) Activity in networks of neuron–like elements. In *Information Theory*, C. Cherry, ed., pp 242–251. Washington, Butterworths.

D. T. Kaplan, J. M.Smith,B. E. H. Saxberg & R. J. Cohen. (1988) Nonlinear dynamics in cardiac conduction. *Math. Biosci.* **90**, 19–48.

A. T. Winfree. (1987) *When Time Breaks Down*, Princeton University Press, Princeton, N.J.
